# Attentional Modulation of Human Pattern Discrimination Psychophysics Reproduced by a Quantitative Model

Laurent Itti, Jochen Braun, Dale K. Lee and Christof Koch
{itti, achim, jjwen, koch}@klab.caltech.edu
Computation & Neural Systems, MSC 139-74
California Institute of Technology, Pasadena, CA 91125, U.S.A.

## Abstract

We previously proposed a quantitative model of early visual processing in primates, based on non-linearly interacting visual filters and statistically efficient decision. We now use this model to interpret the observed modulation of a range of human psychophysical thresholds with and without focal visual attention. Our model – calibrated by an automatic fitting procedure – simultaneously reproduces thresholds for four classical pattern discrimination tasks, performed while attention was engaged by another concurrent task. Our model then predicts that the seemingly complex improvements of certain thresholds, which we observed when attention was fully available for the discrimination tasks, can best be explained by a strengthening of competition among early visual filters.

## 1  INTRODUCTION

What happens when we voluntarily focus our attention to a restricted part of our visual field? Focal attention is often thought as a gating mechanism, which selectively allows a certain spatial location and and certain types of visual features to reach higher visual processes. We here investigate the possibility that attention might have a specific computational modulatory effect on early visual processing.

We and others have observed that focal visual attention can modulate human psychophysical thresholds for simple pattern discrimination tasks [7, 8, 5] When attention is drawn away from a task, for example by "cueing" [12] to another location of the display, or by a second, concurrent task [1, 7, 8], an apparently complex pattern of performance degradation is observed: For some tasks, attention has little or no effect on performance (e.g., detection of luminance increments), while for

other tasks, attention dramatically improves performance (e.g., discrimination of orientation). Our specific findings with dual-task psychophysics are detailed below.

These observations have been paralleled by electrophysiological studies of attention. In the awake macaque, neuronal responses to attended stimuli can be 20% to 100% higher than to otherwise identical unattended stimuli. This has been demonstrated in visual cortical areas V1, V2, and V4 [16, 11, 10, 9] when the animal discriminates stimulus orientation, and in areas MT and MST when the animal discriminates the speed of stimulus motion [17]. Even spontaneous firing rates are 40% larger when attention is directed at a neuron's receptive field [9]. Whether neuronal responses to attended stimuli are merely enhanced [17] or whether they are also more sharply tuned for certain stimulus dimensions [16] remains controversial. Very recently, fMRI studies have shown similar enhancement (as measured with BOLD contrast) in area V1 of humans, specifically at the retinotopic location where subjects had been instructed to focus their attention to [2, 14].

All of these observations directly address the issue of the "top-down" computational effect of attentional focusing onto early visual processing stages. This issue should be distinguished from that of the "bottom-up" control of visual attention [6], which studies which visual features are likely to attract the attention focusing mechanism (e.g., pop-out phenomena and studies of visual search). Top-down attentional modulation happens after attention has been focused to a location of the visual field, and most probably involves the massive feedback circuits which anatomically project from higher cortical areas back to early visual processing areas.

In the present study, we quantify the modulatory effect of attention observed in human psychophysics using a model of early visual processing. The model is based on non-linearly interacting visual filters and statistically efficient decision [4, 5]. Although attention could modulate virtually any visual processing stage (e.g., the decision stage, which compares internal responses from different stimuli), our basic hypothesis here – supported by electrophysiology and fMRI [16, 11, 10, 17, 9, 2, 14] – is that this modulation might happen very early in the visual processing hierarchy. Given this basic hypothesis, we investigate how attention should affect early visual processing in order to quantitatively reproduce the psychophysical results.

## 2 PSYCHOPHYSICAL EXPERIMENTS

We measured attentional modulation of spatial vision thresholds using a dual-task paradigm [15, 7]: At the center of the visual field, a letter discrimination task is presented, while a pattern discrimination task is simultaneously presented at a random peripheral location (4° eccentricity). The central task consists of discriminating between five letters "T" or four "T" and one "L". It has been shown to efficiently engage attention [7]. The peripheral task is chosen from a battery of a classical pattern discrimination tasks, and is the task of interest for this study. Psychophys-

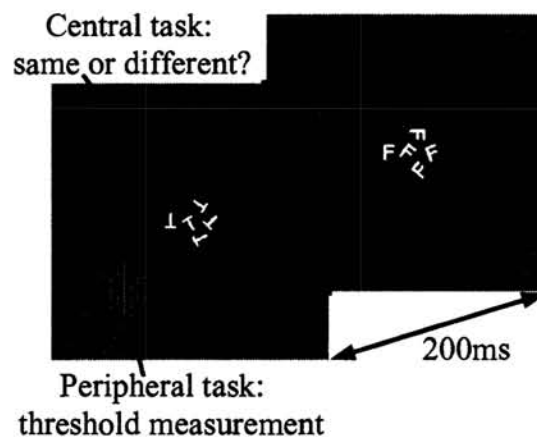

Central task: same or different?

Peripheral task: threshold measurement

ical thresholds are measured for two distinct conditions: In the **"fully attended"** condition, observers are asked to devote their entire attention to the peripheral

task, and to ignore the central task (while still fixating the center of the screen). In the **"poorly attended"** condition, observers are asked to pay full attention to the central task (and the blocks of trials for which performance for the central task falls below a certain cut-off are discarded).

Four classical pattern discrimination tasks were investigated, each with two volunteer subjects (average shown in **Figure 1**), similarly to our previous experiments [7, 8]. Screen luminance resolution was 0.2%. Screen luminance varied from 1 to 90cd/m$^2$ (mean 45cd/m$^2$), room illumination was 5cd/m$^2$ and viewing distance 80cm. The Yes/No (present/absent) paradigm was used (one stimulus presentation per trial). Threshold (75% correct peformance) was reached using a staircase procedure, and computed through a maximum-likelihood fit of a Weibull function with two degrees of freedom to the psychometric curves.

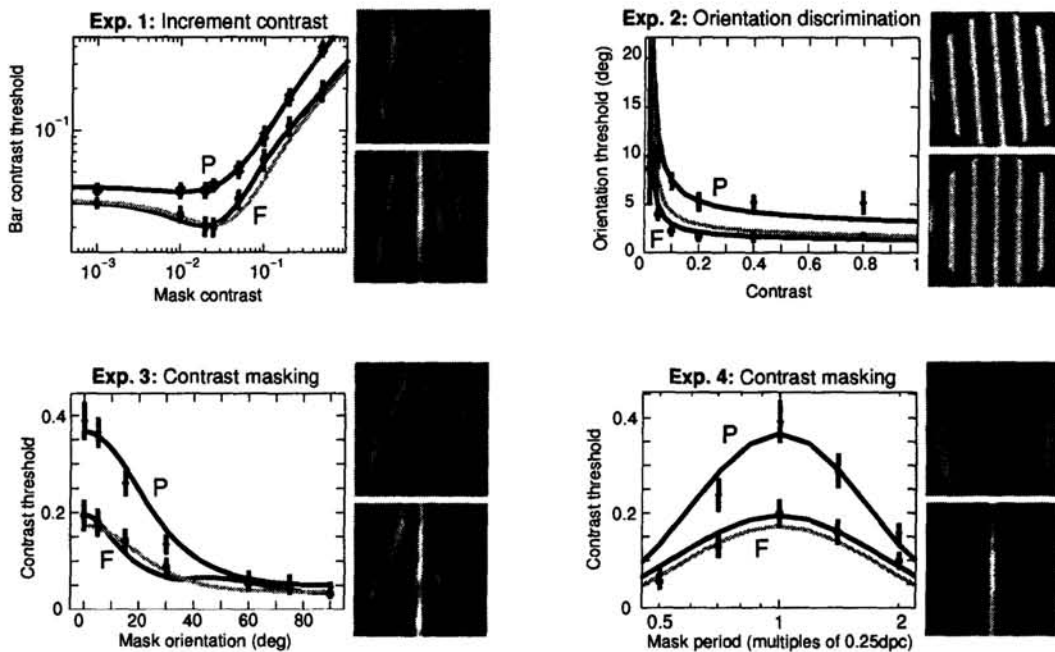

Figure 1: Psychophysical data and model fits using the parameters from Table 1 (P=poorly and F=fully attended). Gray curves: Model predictions for fully attended data, using the poorly attended parameters, except for $\gamma = 2.9$ and $\delta = 2.1$ (see Results).

**Exp. 1** measured increment contrast discrimination threshold: The observer discriminates between a 4cpd (cycles per degree) stochastic oriented mask [7] at fixed contrast, and the same mask plus a low-contrast sixth-derivative-of-Gaussian (D6G) bar; threshold is measured for bar contrast [8]. **Exp. 2** measured orientation discrimination thresholds: The observer discriminates between a vertical and tilted grating at 4cpd; threshold for the angle difference is measured. In addition, two contrast masking tasks were investigated for their sensitivity to non-linearities in visual processing. A 4cpd stochastic mask (50% contrast) was always present, and threshold was measured for the contrast of a vertical superimposed D6G bar. In **Exp. 3**, the orientation of the masker was varied and its spatial frequency fixed (4cpd), while in **Exp. 4** the spatial period of the masker was varied and its orientation vertical. Our aim was to investigate very dissimilar tasks, in particular with respect to the decision strategy used by the observer.

Using the dual-task paradigm, we found mixed attentional effects on psychophysical thresholds, including the appearance of a more pronounced contrast discrimination

"dipper" in **Exp. 1**, substantial improvement of orientation thresholds in **Exp. 2**, and reduced contrast elevations due to masking in **Exps. 3–4** (also see [7, 8]).

# 3   MODEL

The model consists of three successive stages [4, 5]. In the first stage, a bank of Gabor-like linear filters analyzes a fixed location of the visual scene. Here, a single-scale model composed of 12 pairs of filters in quadrature phase, tuned for orientations $\theta \in \Theta$ evenly spanning $180°$, was sufficient to account for the data (although a multi-scale model may account for

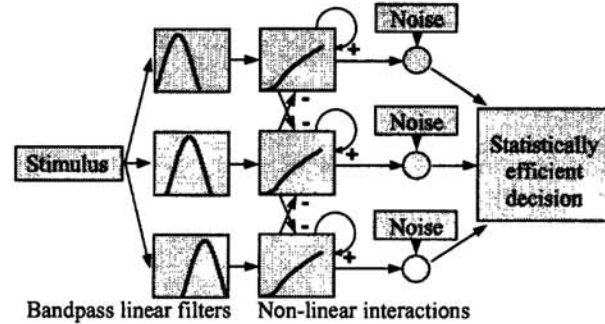

Bandpass linear filters    Non-linear interactions

a wider range of psychophysical thresholds). The linear filters take values between 0.0 and 100.0, then multiplied by a gain factor $A$ (one of the ten free parameters of the model), and to which a small background activity $\epsilon$ is added.

In the second stage, filters non-linearly interact as follows: (1) Each unit receives non-linear self-excitation, and (2) each unit receives non-linear divisive inhibition from a pool of similarly-tuned units: With $E_\theta$ being the linear response from a unit tuned for orientation $\theta$, the pooled response $R_\theta$ is given by:

$$R_\theta = \frac{(E_\theta)^\gamma}{(S)^\delta + \sum_{\theta' \in \Theta} W_\theta(\theta') \, (E_{\theta'})^\delta} + \eta, \qquad \text{where} \qquad W_\theta(\theta') = e^{-\frac{(\theta'-\theta)^2}{2\Sigma_\theta^2}}$$

is a Gaussian weighting function centered around $\theta$, and $\eta$ a positive constant to account for background activity in the pooling stage. This stage is inspired from Heeger's model of gain control in cat V1 [3, 4]. Our formulation, in which none of the parameters is given a particular value, however allows for multiple outcomes, to be determined by fitting the model to our psychophysical data: A sigmoidal ($S > 0, \gamma > \delta$) as well as simple power-law ($S = 0$) or even linear ($\gamma = 1, \delta = 0$) contrast response characteristic could emerge, the responses could be saturating ($\gamma = \delta$) or not ($\gamma \neq \delta$), and the inhibitory pool size ($\Sigma_\theta$) could be broad or narrow. Because striate neurons are noisy, physiological noise is assumed in the model at the outputs of the second stage. The noise level is chosen close to what is typically observed in cortical pyramidal cells, and modeled by Gaussian noise with variance equal to mean taken to some power $\alpha$ determined by fitting.

Because the decision stage – which quantitatively relates activity in the population of pooled noisy units to behavioral discrimination performance – is not fully characterized in humans, we are not in a position to model it in any detail. Instead, we trained our subjects (for 2-3 hours on each task), and assume that they perform close to an "optimal detector". Such optimal detector may be characterized in a formal manner, using Statistical Estimation Theory [4, 5]. We assume that a brain mechanism exists, which, for a given stimulus presentation, builds an internal estimate of some stimulus attribute $\zeta$ (e.g., contrast, orientation, period). The central assumption of our decision stage is that this brain mechanism will perform close to an *unbiased efficient statistic* $T$, which is the best possible estimator of $\zeta$

given the noisy population response from the second stage. The accuracy (variance) with which $T$ estimates $\zeta$ can be computed formally, and is the inverse of the Fisher Information with respect to $\zeta$ [13, 4]. Simply put, this means that, from the first two stages of the model alone, we have a means of computing the best possible estimation performance for $\zeta$, and consequently, the best possible discrimination performance between two stimuli with parameters $\zeta_1$ and $\zeta_2$ [4, 5]. Such statistically efficient decision stage is implementable as a neural network [13].

This decision stage provides a unified framework for optimal discrimination in any behavioral situation, and eliminates the need for task-dependent assumptions about the strategy used by the observers to perform the task in a near optimal manner. Our model allows for a quantitative prediction of human psychophysical thresholds, based on a crude simulation of the physiology of primary visual cortex (area V1).

# 4   RESULTS

All parameters in the model were automatically adjusted in order to best fit the psychophysical data from all experiments. A multidimensional downhill simplex with simulated annealing overhead was used to minimize the root-mean-square distance between the quantitative predictions of the model and the human data [4]. The best-fit parameters obtained independently for the "fully attended" and "poorly attended" conditions are reported in **Table 1**. The model's simultaneous fits to our entire dataset are plotted in **Figure 1** for both conditions. After convergence of the fitting procedure, a measure of how well constrained each parameter was by the data was computed as follows: Each parameter was systematically varied around its best-fit value, in 0.5% steps, and the fitting error was recomputed; the amplitude by which each parameter could be varied before the fitting error increased by more than 10% of its optimum is noted as a standard deviation in **Table 1**. A lower deviation indicates that the parameter is more strongly constrained by the dataset.

**Table 1. Model parameters for both attentional conditions.**

| Name | Symbol | fully attended | poorly attended |
|---|---|---|---|
| Linear gain[†] | $A$ | $1.7 \pm 0.2$ | $8.2 \pm 0.9$ |
| Activity-independent inhibition[†] | $S$ | $14.1 \pm 2.3$ | $101.5 \pm 16.6$ |
| Excitatory exponent | $\gamma$ | $3.36 \pm 0.02$ | $2.09 \pm 0.01$ |
| Inhibitory exponent | $\delta$ | $2.48 \pm 0.02$ | $1.51 \pm 0.02$ |
| Noise exponent | $\alpha$ | $1.34 \pm 0.07$ | $1.39 \pm 0.08$ |
| Background activity, linear stage | $\epsilon$ | $1.13 \pm 0.35$ | $1.25 \pm 0.60$ |
| Background activity, pooling stage | $\eta$ | $0.18 \pm 0.05$ | $0.77 \pm 0.11$ |
| Spatial period tuning width[×] | $\sigma_\lambda$ | $0.85 \pm 0.06$ oct. | $0.85 \pm 0.09$ oct. |
| Orientation tuning width[×] | $\sigma_\theta$ | $26° \pm 2.4°$ | $38° \pm 5.5°$ |
| Orientation pooling width[×] | $\Sigma_\theta$ | $48° \pm 25°$ | $50° \pm 26°$ |

† Dynamic range of linear filters is $[\epsilon \ldots 100.0 \times A + \epsilon]$.

× For clarity, FWHM values are given rather than $\sigma$ values (FWHM $= 2\sigma\sqrt{2\ln(2)}$).

Although no human bias was introduced during the fitting procedure, interestingly, all of the model's internal parameters reached physiologically plausible best-fit values, such as, for example, slightly supra-Poisson noise level ($\alpha \approx 1.35$), $\approx 30°$ orientation tuning FWHM (full-width at half-maximum), and $\approx 0.85$ octave spatial period tuning FWHM. Some of the internal characteristics of the model which more closely relate to the putative underlying physiological mechanisms are shown in **Figure 2**.

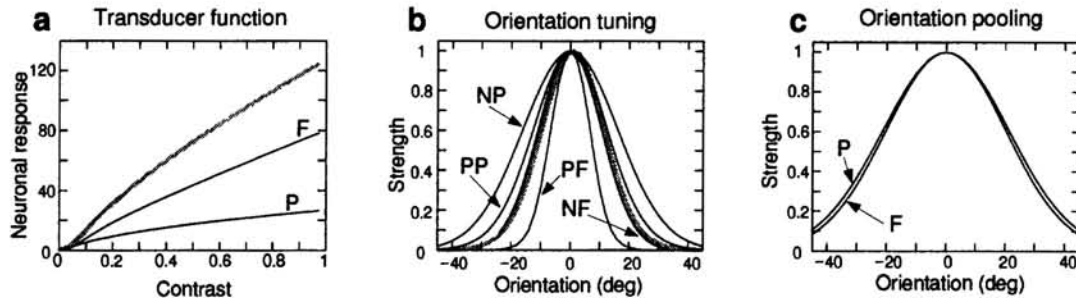

Figure 2: Internals of the model. **(a)** The response function of individual units to contrast was sigmoidal under full (F) and almost linear under poor (P) attention. **(b)** Native linear orientation tuning was broader under poor (NP) than full (NF) attention, but it was sharpened in both cases by pooling (PP=pooled poor, and PF=pooled full attention). **(c)** There was no difference in orientation pooling width under poor (P) or full (F) attention. Using poorly attended parameters, except for $\gamma = 2.9$ and $\delta = 2.1$ (grey curves), yielded steep non-linear contrast response, and intermediary tuning (same width as NF).

In **Table 1**, attention had the following significant effects on the model's parameters: 1) Both pooling exponents $(\gamma, \delta)$ were higher; 2) the tuning width $(\sigma_\theta)$ was narrower; 3) the linear gain $(A)$ and associated activity-independent inhibition $(S)$ were lower; and 4) the background activity of the pooling stage was lower. This yielded increased competition between filters: The network behaved more like a winner-take-all under full attention, and more like a linear network of independent units under poor attention. While the attentional modulation of $\gamma, \delta$ and $\sigma_\theta$ are easy to interpret, its effect on the $A, S$ and $\eta$ is more difficult to understand.

Consequently, we conducted a further automatic fit, which, starting from the "poorly attended" parameters, was only allowed to alter $\gamma$ and $\delta$ to fit the "fully attended" data. The motivation for not varying $\sigma_\theta$ was that we observed significant sharpening of the tuning induced by higher exponents $\gamma, \delta$ **(Figure 2)**. Also, slight changes in the difference $\gamma - \delta$ can easily produce large changes in the overall gain of the system, hence compensating for changes in $A, S$ and $\eta$. (We however do not imply here that $\sigma_\theta, A, S$ and $\eta$ are redundant parameters; there is only a small range around the best-fit point over which $\gamma$ and $\delta$ can compensate for variations in the other parameters, without dramatically impairing the quality of fit).

Although the new fit was not as accurate as that obtained with all parameters allowed to vary, it appeared that a simple modification of the pooling exponents well captured the effect of attention **(Figure 1)**. Hence, the "poorly attended" parameters of **Table 1** well described the "poorly attended" data, and the same parameters except for $\gamma = 2.9$ and $\delta = 2.1$ well described the "fully attended" data.

A variety of other simple parameter modifications were also tested, but none except for the pooling exponents $(\gamma, \delta)$ could fully account for the attentional modulation. These modifications include: Changes in gain (obtained by modifying $A$ only, $\gamma$ only, or $\delta$ only), in tuning $(\sigma_\theta)$, in the extent of the inhibitory pool $(\Sigma_\theta)$, and in the noise level $(\alpha)$. A more systematic study, in which all possible parameter subsets are successively examined, is currently in progress in our laboratory.

# 5   DISCUSSION and CONCLUSION

At the basis of our results is the hypothesis that attention might modulate the earlier rather than the later stages of visual processing. We found that a very

simple, prototypical, task-independent enhancement of the amount of competition between early visual filters accounts well for the human data. This enhancement resulted from increases in parameters $\gamma$ and $\delta$ in the model, and was paralleled by an increase in contrast gain and a sharpening in orientation tuning. Although it is not possible from our data to rule out any attentional modulation at later stages, our hypothesis has recently received experimental support that attention indeed modulates early visual processing in humans [2, 14].

More psychophysical experiments are needed to investigate attentional modulation at later processing stages. For example, it might be possible to study the effect of attention on the decision stage by manipulating attention during experiments involving decision uncertainty. In the absence of such results, we have attempted in our experiments to minimize the possible impact of attention on later stages, by using only simple stimulus patterns devoid of conceptual or emotional meaning, such as to involve as little as possible the more cognitive stages of visual processing.

Our finding that attention may increase the amount of competition between early visual filters is accompanied by an enhancement of the gain and sensitivity of the filters, and by a sharpening of their tuning properties. The existence of two such processing states – one, more sensitive and selective inside the focus of attention, and the other, more broadly-tuned and non-specific outside – can be justified by at least two observations: First, the higher level of activity in attended neurons consumes more energy, which may not be desirable over the entire extent of visual cortices. Second, although less efficient for fine discriminations, the broadly-tuned and non-specific state may have greater ability at catching unexpected, non-specific visual events. In this perspective, this state would be desirable as an input to bottom-up, visual alerting mechanisms, which monitor the rest of our visual world while we are focusing on a specific task requiring high focal accuracy.

## Acknowledgements

This research was supported by ONR and NSF (Caltech ERC).

## References

[1] Bonnel AM, Stein JF, Bertucci P. *Q J Exp Psychol [A]* 1992;44(4):601-26
[2] Gandhi SP, Heeger DJ, Boynton GM. *Inv Opht Vis Sci (ARVO'98)* 1998;39(4):5194
[3] Heeger DJ. *Vis Neurosci* 1992;9:181-97
[4] Itti L, Braun J, Lee DK, Koch C. *Proc NIPS*97* (in press)
[5] Itti L, Koch C, Braun J. *Inv Opht Vis Sci (Proc ARVO'98)* 1998;39(4):2934
[6] Koch C, Ullman S. *Hum Neurobiol* 1985;4:219-27
[7] Lee DK, Koch C, Braun J. *Vis Res* 1997:37(17):2409-18
[8] Lee DK, Koch C, Itti L, Braun J. *Inv Opht Vis Sci (Proc ARVO'98)* 1998;39(4):2938
[9] Luck SJ, Chelazzi L, Hillyard SA, Desimone R. *J Neurophysiol* 1997;77(1):24-42
[10] Maunsell JH. *Science* 1995;270(5237)764-9
[11] Motter BC. *J Neurophysiol* 1993;70(3):909-19
[12] Nakayama K, Mackeben M. *Vis Res* 1989;29(11):1631-47
[13] Pouget A, Zhang K, Deneve S, Latham PE. *Neur Comp* 1998;10:373-401
[14] Somers DC, *et al. Inv Opht Vis Sci (Proc ARVO'98)* 1998;39(4):5192
[15] Sperling G, Melchner MJ. *Science* 1978;202:315-8
[16] Spitzer H, Desimone R, Moran J. *Science* 1988;240(4850):338-40
[17] Treue S, Maunsell JH. *Nature* 1996;382(6591):539-41